# Sparse recovery by thresholded non-negative least squares

**Martin Slawski and Matthias Hein**
Department of Computer Science
Saarland University
Campus E 1.1, Saarbrücken, Germany
{ms,hein}@cs.uni-saarland.de

## Abstract

Non-negative data are commonly encountered in numerous fields, making non-negative least squares regression (NNLS) a frequently used tool. At least relative to its simplicity, it often performs rather well in practice. Serious doubts about its usefulness arise for modern high-dimensional linear models. Even in this setting − unlike first intuition may suggest − we show that for a broad class of designs, NNLS is resistant to overfitting and works excellently for sparse recovery when combined with thresholding, experimentally even outperforming $\ell_1$-regularization. Since NNLS also circumvents the delicate choice of a regularization parameter, our findings suggest that NNLS may be the method of choice.

## 1   Introduction

Consider the linear regression model

$$y = X\beta^* + \varepsilon, \tag{1}$$

where $y$ is a vector of observations, $X \in \mathbb{R}^{n \times p}$ a design matrix, $\varepsilon$ a vector of noise and $\beta^*$ a vector of coefficients to be estimated. Throughout this paper, we are concerned with a high-dimensional setting in which the number of unknowns $p$ is at least of the same order of magnitude as the number of observations $n$, i.e. $p = O(n)$ or even $p \gg n$, in which case one cannot hope to recover the target $\beta^*$ if it does not satisfy one of various kinds of sparsity constraints, the simplest being that $\beta^*$ is supported on $S = \{j : \beta_j^* \neq 0\}$, $|S| = s < n$. In this paper, we additionally assume that $\beta^*$ is non-negative, i.e. $\beta^* \in \mathbb{R}_+^p$. This constraint is particularly relevant, since non-negative data occur frequently, e.g. in the form pixel intensity values of an image, time measurements, histograms or count data, economical quantities such as prices, incomes and growth rates. Non-negativity constraints emerge in numerous deconvolution and unmixing problems in diverse fields such as acoustics [1], astronomical imaging [2], computer vision [3], genomics [4], proteomics [5] and spectroscopy [6]; see [7] for a survey. Sparse recovery of non-negative signals in a noiseless setting ($\varepsilon = 0$) has been studied in a series of recent papers [8, 9, 10, 11]. One important finding of this body of work is that non-negativity constraints alone may suffice for sparse recovery, without the need to employ sparsity-promoting $\ell_1$-regularization as usually. The main contribution of the present paper is a transfer of this intriguing result to a more realistic noisy setup, contradicting the well-established paradigm that regularized estimation is necessary to cope with high dimensionality and to prevent over-adaptation to noise. More specifically, we study non-negative least squares (NNLS)

$$\min_{\beta \succeq 0} \frac{1}{n} \|y - X\beta\|_2^2 \tag{2}$$

with minimizer $\widehat{\beta}$ and its counterpart after hard thresholding $\widehat{\beta}(\lambda)$,

$$\widehat{\beta}_j(\lambda) = \begin{cases} \widehat{\beta}_j, & \widehat{\beta}_j > \lambda, \\ 0, & \text{otherwise}, \ j = 1, \ldots, p, \end{cases} \tag{3}$$

where $\lambda \geq 0$ is a threshold, and state conditions under which it is possible to infer the support $S$ by $\widehat{S}(\lambda) = \{j : \widehat{\beta}_j(\lambda) > 0\}$. Classical work on the problem [12] gives a positive answer for fixed $p$, while in case one follows the modern statistical trend, one would add a regularizer to (2) in order to encourage sparsity: the most popular approach is $\ell_1$-regularized least squares (lasso, [13]), which is easy to implement and comes with strong theoretical guarantees with regard to prediction and estimation of $\beta^*$ in the $\ell_2$-norm over a broad range of designs (see [14] for a review). On the other hand, the rather restrictive 'irrepresentable condition' on the design is essentially necessary in order to infer the support $S$ from the sparsity pattern of the lasso [15, 16]. In view of its tendency to assign non-zero weights to elements of the off-support $S^c = \{1, \ldots, p\} \setminus S$, several researchers, e.g. [17, 18, 19], suggest to apply hard thresholding to the lasso solution to achieve support recovery. In light of this, thresholding a non-negative least squares solution, provided it is close to the target w.r.t. the $\ell_\infty$-norm, is more attractive for at least two reasons: first, there is no need to carefully tune the amount of $\ell_1$-regularization prior to thresholding; second, one may hope to detect relatively small non-zero coefficients whose recovery is negatively affected by the bias of $\ell_1$-regularization.

**Outline.** We first prove a bound on the mean square prediction error of the NNLS estimator, demonstrating that it may be resistant to overfitting. Section 3 contains our main results on sparse recovery with noise. Experiments providing strong support of our theoretical findings are presented in Section 4. Most of the proofs as well as technical definitions are relegated to the supplement.

**Notation.** Let $J, K$ be index sets. For a matrix $A \in \mathbb{R}^{n \times m}$, $A_J$ denotes the matrix one obtains by extracting the columns corresponding to $J$. For $j = 1, \ldots, m$, $A_j$ denotes the $j$-th column of $A$. The matrix $A_{JK}$ is the sub-matrix of $A$ by extracting rows in $J$ and columns in $K$. For $v \in \mathbb{R}^m$, $v_J$ is the sub-vector corresponding to $J$. The identity matrix is denoted by $I$ and vectors of ones by $\mathbf{1}$. The symbols $\preceq (\prec), \succeq (\succ)$ denote entry-wise (strict) inequalities. Lower and uppercase c's denote positive universal constants (not depending on $n, p, s$) whose values may differ from line to line.

**Assumptions.** We here fix what is assumed throughout the paper unless stated otherwise. Model (1) is assumed to hold. The matrix $X$ is assumed to be non-random and scaled s.t. $\|X_j\|_2^2 = n \, \forall j$. We assume that $\varepsilon$ has i.i.d. zero-mean sub-Gaussian entries with parameter $\sigma > 0$, cf. supplement.

## 2 Prediction error and uniqueness of the solution

In the following, the quantity of interest is the mean squared prediction error (MSE) $\frac{1}{n}\|X\beta^* - X\widehat{\beta}\|_2^2$.

**NNLS does not necessarily overfit.** It is well-known that the MSE of ordinary least squares (OLS) as well as that of ridge regression in general does not vanish unless $p/n \to 0$. Can one do better with non-negativity constraints ? Obviously, the answer is negative for general $X$. To make this clear, let a design matrix $\widetilde{X}$ be given and set $X = [\widetilde{X} - \widetilde{X}]$ by concatenating $\widetilde{X}$ and $-\widetilde{X}$ columnwise. The non-negativity constraint is then vacuous in the sense that $X\widehat{\beta} = X\widehat{\beta}^{\text{ols}}$, where $\widehat{\beta}^{\text{ols}}$ is any OLS solution. However, non-negativity constraints on $\beta$ can be strong when coupled with the following condition imposed on the Gram matrix $\Sigma = \frac{1}{n}X^\top X$.

**Self-regularizing property.** We call a design self-regularizing with universal constant $\kappa \in (0, 1]$ if

$$\beta^\top \Sigma \beta \geq \kappa (\mathbf{1}^\top \beta)^2 \quad \forall \beta \succeq 0. \tag{4}$$

The term 'self-regularizing' refers to the fact that the quadratic form in $\Sigma$ restricted to the non-negative orthant acts like a regularizer arising from the design itself. Let us consider two examples:
(1) If $\Sigma \succeq \kappa_0 > 0$, i.e. all entries of the Gram matrix are at least $\kappa_0$, then (4) holds with $\kappa = \kappa_0$.
(2) If the Gram matrix is entry-wise non-negative and if the set of predictors indexed by $\{1, \ldots, p\}$ can be partitioned into subsets $B_1, \ldots, B_\mathcal{B}$ such that $\min_{1 \leq b \leq \mathcal{B}} \frac{1}{n} X_{B_b}^\top X_{B_b} \succeq \kappa_0$, then

$$\min_{\beta \succeq 0} \beta^\top \Sigma \beta \geq \sum_{b=1}^{\mathcal{B}} \beta_{B_b}^\top \frac{1}{n} X_{B_b}^\top X_{B_b} \beta_{B_b} \geq \kappa_0 \sum_{b=1}^{\mathcal{B}} (\mathbf{1}^\top \beta_{B_b})^2 \geq \frac{\kappa_0}{\mathcal{B}} (\mathbf{1}^\top \beta)^2.$$

In particular, this applies to design matrices whose entries $X_{ij} = \phi_j(u_i)$ contain the function evaluations of non-negative functions $\{\phi_j\}_{j=1}^p$ traditionally used for data smoothing such as splines, Gaussians and related 'localized' functions at points $\{u_i\}_{i=1}^n$ in some fixed interval, see Figure 1.

For self-regularizing designs, the MSE of NNLS can be controlled as follows.

**Theorem 1.** *Let $\Sigma$ fulfill the self-regularizing property with constant $\kappa$. Then, with probability no less than 1 - 2/p, the NNLS estimator obeys*

$$\frac{1}{n}\|X\beta^* - X\widehat{\beta}\|_2^2 \leq \frac{8\sigma}{\kappa}\sqrt{\frac{2\log p}{n}}\|\beta^*\|_1 + \frac{8\sigma^2}{\kappa}\frac{\log p}{n}.$$

The statement implies that for self-regularizing designs, NNLS is consistent in the sense that its MSE, which is of the order $O(\sqrt{\log(p)/n}\,\|\beta^*\|_1)$, may vanish as $n \to \infty$ even if the number of predictors $p$ scales up to sub-exponentially in $n$. It is important to note that exact sparsity of $\beta^*$ is not needed for Theorem 1 to hold. The rate is the same as for the lasso if no further assumptions on the design are made, a result that is essentially obtained in the pioneering work [20].

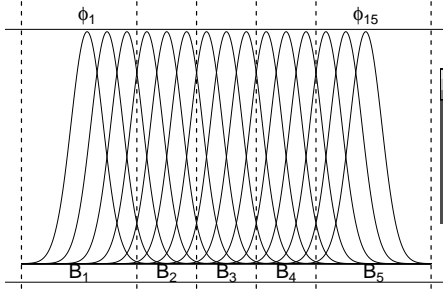
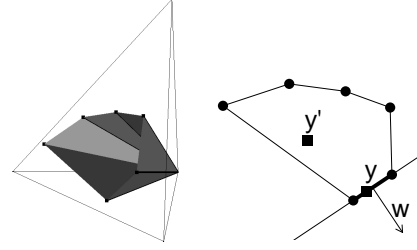

Figure 1: Block partitioning of 15 Gaussians into $\mathcal{B} = 5$ blocks. The right part shows the corresponding pattern of the Gram matrix.

Figure 2: A polyhedral cone in $\mathbb{R}^3$ and its intersection with the simplex (right). The point $y$ is contained in a face (bold) with normal vector $w$, whereas $y'$ is not.

**Uniqueness of the solution.** Considerable insight can be gained by looking at the NNLS problem (2) from the perspective of convex geometry. Denote by $\mathcal{C} = X\mathbb{R}_+^p$ the polyhedral cone generated by the columns $\{X_j\}_{j=1}^p$ of $X$, which are henceforth assumed to be in general position in $\mathbb{R}^n$. As visualized in Figure 2, sparse recovery by non-negativity constraints can be analyzed by studying the face lattice of $\mathcal{C}$ [9, 10, 11]. For $F \subseteq \{1, \ldots, p\}$, we say that $X_F\mathbb{R}_+^{|F|}$ is a face of $\mathcal{C}$ if there exists a separating hyperplane with normal vector $w$ passing through the origin such that $\langle X_j, w\rangle > 0$, $j \notin F$, $\langle X_j, w\rangle = 0$, $j \in F$. Sparse recovery in a noiseless setting ($\varepsilon = 0$) can then be characterized concisely by the following statement which can essentially be found in prior work [9, 10, 11, 21].

**Proposition 1.** *Let $y = X\beta^*$, where $\beta^* \succeq 0$ has support $S$, $|S| = s$. If $X_S\mathbb{R}_+^s$ is a face of $\mathcal{C}$ and the columns of $X$ are in general position in $\mathbb{R}^n$, then the constrained linear system $X\beta = y$ sb.t. $\beta \succeq 0$, has $\beta^*$ as its unique solution.*

*Proof.* By definition, since $X_S\mathbb{R}_+^s$ is a face of $\mathcal{C}$, there exists a $w \in \mathbb{R}^n$ s.t. $\langle X_j, w\rangle = 0$, $j \in S$, $\langle X_j, w\rangle > 0$, $j \in S^c$. Assume that there is a second solution $\beta^* + \delta$, $\delta \neq 0$. Expand $X_S(\beta_S^* + \delta_S) + X_{S^c}\delta_{S^c} = y$. Multiplying both sides by $w^\top$ yields $\sum_{j \in S^c}\langle X_j, w\rangle\delta_j = 0$. Since $\beta_{S^c}^* = 0$, feasibility requires $\delta_j \geq 0$, $j \in S^c$. All inner products within the sum are positive, concluding that $\delta_{S^c} = 0$. General position implies $\delta_S = 0$. $\square$

Given Theorem 1 and Proposition 1, we turn to uniqueness in the noisy case.

**Corollary 1.** *In the setting of Theorem 1, if $\|\beta^*\|_1 = o(\sqrt{n/\log(p)})$, then the NNLS solution $\widehat{\beta}$ is unique with high probability.*

*Proof.* Suppose first that $y \notin \mathcal{C} = X\mathbb{R}_+^p$, then $X\widehat{\beta}$, the projection of $y$ on $\mathcal{C}$, is contained in its boundary, i.e. in a lower-dimensional face. Using general position of the columns of $X$, Proposition 1 implies that $\widehat{\beta}$ is unique. If $y$ were already contained in $\mathcal{C}$, one would have $y = X\widehat{\beta}$ and hence

$$\frac{1}{n}\|X\beta^* - X\widehat{\beta}\|_2^2 = \frac{1}{n}\|X\beta^* - y\|_2^2 = \frac{1}{n}\|\varepsilon\|_2^2 = O(1), \text{ with high probability,} \qquad (5)$$

using concentration of measure of the norm of the sub-Gaussian random vector $\varepsilon$. With the assumed scaling for $\|\beta^*\|_1$, $\frac{1}{n}\|X\beta^* - X\widehat{\beta}\|_2^2 = o(1)$ in view of Theorem 1, which contradicts (5). $\square$

## 3 Sparse recovery in the presence of noise

Proposition 1 states that support recovery requires $X_S \mathbb{R}_+^s$ to be a face of $X\mathbb{R}_+^p$, which is equivalent to the existence of a hyperplane separating $X_S \mathbb{R}_+^s$ from the rest of $\mathcal{C}$. For the noisy case, mere separation is not enough — a quantification is needed, which is provided by the following two incoherence constants that are of central importance for our main result. Both are specific to NNLS and have not been used previously in the literature on sparse recovery.

**Definition 1.** *For some fixed $S \subset \{1, \ldots, p\}$, the separating hyperplane constant is defined as*

$$\widehat{\tau}(S) = \max_{\tau, w} \tau$$

$$sb.t. \quad \frac{1}{\sqrt{n}} X_S^\top w = 0, \quad \frac{1}{\sqrt{n}} X_{S^c}^\top w \succeq \tau \mathbf{1}, \quad \|w\|_2 \le 1, \tag{6}$$

$$\overset{duality}{=} \min_{\theta \in \mathbb{R}^s, \, \lambda \in T^{p-s-1}} \frac{1}{\sqrt{n}} \|X_S \theta - X_{S^c} \lambda\|_2, \tag{7}$$

where $T^{m-1} = \{v \in \mathbb{R}^m : v \succeq 0, \mathbf{1}^\top v = 1\}$ denotes the simplex in $\mathbb{R}^m$, i.e. $\widehat{\tau}(S)$ equals the distance of the subspace spanned by $\{X_j\}_{j \in S}$ and the convex hull of $\{X_j\}_{j \in S^c}$.
We denote by $\Pi_S$ and $\Pi_S^\perp$ the orthogonal projections on the subspace spanned by $\{X_j\}_{j \in S}$ and its orthogonal complement, respectively, and set $Z = \Pi_S^\perp X_{S^c}$. One can equivalently express (7) as

$$\widehat{\tau}^2(S) = \min_{\lambda \in T^{p-s-1}} \lambda^\top \frac{1}{n} Z^\top Z \lambda. \tag{8}$$

The second incoherence constant we need can be traced back to the KKT optimality conditions of the NNLS problem. The role of the following quantity is best understood from (13) below.

**Definition 2.** *For some fixed $S \subset \{1, \ldots, p\}$ and $Z = \Pi_S^\perp X_{S^c}$, $\widehat{\omega}(S)$ is defined as*

$$\widehat{\omega}(S) = \min_{\emptyset \ne F \subseteq \{1, \ldots, p-s\}} \min_{v \in \mathcal{V}(F)} \left\| \frac{1}{n} Z_F^\top Z_F v \right\|_\infty, \quad \mathcal{V}(F) = \{v \in \mathbb{R}^{|F|} : \|v\|_\infty = 1, \, v \succeq 0\}. \tag{9}$$

In the supplement, we show that i) $\widehat{\omega}(S) > 0 \Leftrightarrow \widehat{\tau}(S) > 0 \Leftrightarrow X_S \mathbb{R}_+^s$ is a face of $\mathcal{C}$, and ii) $\widehat{\omega}(S) \le 1$, with equality if $\{X_j\}_{j \in S}$ and $\{X_j\}_{j \in S^c}$ are orthogonal and $\frac{1}{n} X_{S^c}^\top X_{S^c}$ is entry-wise non-negative. Denoting the entries of $\Sigma = \frac{1}{n} X^\top X$ by $\sigma_{jk}$, $1 \le j, k \le p$, our main result additionally involves the constants

$$\mu(S) = \max_{j \in S} \max_{k \in S^c} |\sigma_{jk}|, \quad \mu_+(S) = \max_{j \in S} \sum_{k \in S^c} |\sigma_{jk}|, \quad \beta_{\min}(S) = \min_{j \in S} \beta_j^*,$$
$$K(S) = \max_{v: \|v\|_\infty = 1} \left\| \Sigma_{SS}^{-1} v \right\|_\infty, \quad \phi_{\min}(S) = \min_{v: \|v\|_2 = 1} \|\Sigma_{SS} v\|_2. \tag{10}$$

**Theorem 2.** *Consider the thresholded NNLS estimator $\widehat{\beta}(\lambda)$ defined in (3) with support $\widehat{S}(\lambda)$.*
*(i) If $\lambda > \frac{2\sigma}{\widehat{\tau}^2(S)} \sqrt{\frac{2 \log p}{n}}$ and*

$$\beta_{\min}(S) > \widetilde{\lambda}, \quad \widetilde{\lambda} = \lambda(1 + K(S)\mu(S)) + \frac{2\sigma}{\{\phi_{\min}(S)\}^{1/2}} \sqrt{\frac{2 \log p}{n}},$$

*(ii) or if $\lambda > \frac{2\sigma}{\widehat{\omega}(S)} \sqrt{\frac{2 \log p}{n}}$ and*

$$\beta_{\min}(S) > \widetilde{\lambda}, \quad \widetilde{\lambda} = \lambda(1 + K(S)\mu_+(S)) + \frac{2\sigma}{\{\phi_{\min}(S)\}^{1/2}} \sqrt{\frac{2 \log p}{n}},$$

*then $\|\widehat{\beta}(\lambda) - \beta^*\|_\infty \le \widetilde{\lambda}$ and $\widehat{S}(\lambda) = S$ with probability no less than $1 - 10/p$.*

**Remark.** The concept of a separating functional as in (6) is also used to show support recovery for the lasso [15, 16] as well as for orthogonal matching pursuit [22, 23]. The 'irrepresentable condition' employed in these works requires the existence of a separation constant $\gamma(S) > 0$ such that

$$\max_{j \in S^c} |X_j^\top X_S (X_S^\top X_S)^{-1} \operatorname{sign}(\beta_S^*)| \le 1 - \gamma(S), \quad \text{while } |X_j^\top X_S (X_S^\top X_S)^{-1} \operatorname{sign}(\beta_S^*)| = 1, \, j \in S,$$

hence $\{X_j\}_{j \in S}$ and $\{X_j\}_{j \in S^c}$ are separated by the functional $|\langle \cdot, X_S (X_S^\top X_S)^{-1} \operatorname{sign}(\beta_S^*) \rangle|$.

In order to prove Theorem 2, we need two lemmas first. The first one is immediate from the KKT optimality conditions of the NNLS problem.

**Lemma 1.** $\widehat{\beta}$ *is a minimizer of* (2) *if and only if there exists* $F \subseteq \{1, \ldots, p\}$ *such that*

$$\frac{1}{n} X_j^\top (y - X\widehat{\beta}) = 0, \text{ and } \widehat{\beta}_j > 0, \ j \in F, \quad \frac{1}{n} X_j^\top (y - X\widehat{\beta}) \le 0, \text{ and } \widehat{\beta}_j = 0, \ j \in F^c.$$

The next lemma is crucial, since it permits us to decouple $\widehat{\beta}_S$ from $\widehat{\beta}_{S^c}$.

**Lemma 2.** *Consider the two non-negative least squares problems*

$$(P1): \min_{\beta^{(P1)} \succeq 0} \frac{1}{n} \|\Pi_S^\perp (\varepsilon - X_{S^c} \beta^{(P1)})\|_2^2 \quad (P2): \min_{\beta^{(P2)} \succeq 0} \frac{1}{n} \|\Pi_S y - X_S \beta^{(P2)} - \Pi_S X_{S^c} \widehat{\beta}^{(P1)}\|_2^2$$

*with minimizers* $\widehat{\beta}^{(P1)}$ *of* $(P1)$ *and* $\widehat{\beta}^{(P2)}$ *of* $(P2)$*, respectively. If* $\widehat{\beta}^{(P2)} \succ 0$*, then setting* $\widehat{\beta}_S = \widehat{\beta}^{(P2)}$ *and* $\widehat{\beta}_{S^c} = \widehat{\beta}^{(P1)}$ *yields a minimizer* $\widehat{\beta}$ *of the non-negative least squares problem* (2).

*Proof of Theorem 2.* The proofs of parts (i) and (ii) overlap to a large extent. Steps specific to one of the two parts are preceded by '(i)' or '(ii)'. Consider problem $(P1)$ of Lemma 2.

**Step 1:** *Controlling* $\|\widehat{\beta}^{(P1)}\|_1$ *via* $\widehat{\tau}^2(S)$*, controlling* $\|\widehat{\beta}^{(P1)}\|_\infty$ *via* $\widehat{\omega}(S)$*.*

(i) With $\xi = \Pi_S^\perp \varepsilon$, since $\widehat{\beta}^{(P1)}$ is a minimizer, it satisfies

$$\frac{1}{n} \|\xi - Z\widehat{\beta}^{(P1)}\|_2^2 \le \frac{1}{n} \|\xi\|_2^2 \ \Rightarrow \ (\widehat{\beta}^{(P1)})^\top \frac{1}{n} Z^\top Z \widehat{\beta}^{(P1)} \le \|\widehat{\beta}^{(P1)}\|_1 M, \ M = \max_{1 \le j \le (p-s)} \frac{2}{n} |Z_j^\top \xi|. \tag{11}$$

As observed in (8), $\widehat{\tau}^2(S) = \min_{\lambda \in T^{p-s-1}} \lambda^\top \frac{1}{n} Z^\top Z \lambda$, s.t. the l.h.s. can be lower bounded via

$$(\widehat{\beta}^{(P1)})^\top \frac{1}{n} Z^\top Z \widehat{\beta}^{(P1)} \ge \left\{ \min_{\lambda \in T^{p-s-1}} \lambda^\top \frac{1}{n} Z^\top Z \lambda \right\} \|\widehat{\beta}^{(P1)}\|_1^2 = \widehat{\tau}^2(S) \|\widehat{\beta}^{(P1)}\|_1^2. \tag{12}$$

Combining (11) and (12), we have $\|\widehat{\beta}^{(P1)}\|_1 \le \frac{1}{\widehat{\tau}^2(S)} M$.

(ii) In view of Lemma 1, there exists a set $F \subseteq \{1, \ldots, p-s\}$ (we may assume $F \ne \emptyset$, otherwise $\widehat{\beta}^{(P1)} = 0$) such that $\widehat{\beta}_{F^c}^{(P1)} = 0$ and such that

$$\frac{1}{n} Z_F^\top Z_F \widehat{\beta}_F^{(P1)} = \frac{2}{n} Z_F^\top \xi, \ \Rightarrow \ \left\| \frac{1}{n} Z_F^\top Z_F \widehat{\beta}_F^{(P1)} \right\|_\infty = \left\| \frac{2}{n} Z_F^\top \xi \right\|_\infty$$

$$\Rightarrow \ \min_{v \in \mathcal{V}(F)} \left\| \frac{1}{n} Z_F^\top Z_F v \right\|_\infty \|\widehat{\beta}^{(P1)}\|_\infty \le \left\| \frac{2}{n} Z^\top \xi \right\|_\infty, \ \mathcal{V}(F) = \{v \in \mathbb{R}^{|F|} : \ \|v\|_\infty = 1, \ v \succeq 0\}$$

$$\Rightarrow \ \widehat{\omega}(S) \|\widehat{\beta}^{(P1)}\|_\infty = \min_{\emptyset \ne F \subseteq \{1,\ldots,p-s\}} \min_{v \in \mathcal{V}(F)} \left\| \frac{1}{n} Z_F Z_F v \right\|_\infty \|\widehat{\beta}^{(P1)}\|_\infty \le \left\| \frac{2}{n} Z^\top \xi \right\|_\infty = M, \tag{13}$$

where we have used Definition 2. We conclude that $\|\widehat{\beta}^{(P1)}\|_\infty \le \frac{M}{\widehat{\omega}(S)}$.

**Step 2:** *Back-substitution into (P2).* Equipped with the bounds just derived, we insert $\widehat{\beta}^{(P1)}$ into problem $(P2)$ of Lemma 2, and show that in conjunction with the assumptions made for the minimum support coefficient $\beta_{\min}(S)$, the *ordinary* least squares estimator corresponding to $(P2)$

$$\bar{\beta}^{(P2)} = \operatorname*{argmin}_{\beta^{(P2)}} \frac{1}{n} \|\Pi_S y - X_S \beta^{(P2)} - \Pi_S X_{S^c} \widehat{\beta}^{(P1)}\|_2^2$$

has only positive components. Lemma 2 then yields $\bar{\beta}^{(P2)} = \widehat{\beta}^{(P2)} = \widehat{\beta}_S$. Using the closed form expression for the ordinary least squares estimator, one obtains

$$\bar{\beta}^{(P2)} = \frac{1}{n} \Sigma_{SS}^{-1} X_S^\top (X_S \beta_S^* + \Pi_S \varepsilon - \Pi_S X_{S^c} \widehat{\beta}^{(P1)}) = \beta_S^* + \frac{1}{n} \Sigma_{SS}^{-1} X_S^\top \varepsilon - \Sigma_{SS}^{-1} \Sigma_{SS^c} \widehat{\beta}^{(P1)}.$$

It remains to control the deviation terms $\overline{M} = \|\frac{1}{n} \Sigma_{SS}^{-1} X_S^\top \varepsilon\|_\infty$ and $\|\Sigma_{SS}^{-1} \Sigma_{SS^c} \widehat{\beta}^{(P1)}\|_\infty$. We have

$$\|\Sigma_{SS}^{-1} \Sigma_{SS^c} \widehat{\beta}^{(P1)}\|_\infty \le \max_{v: \|v\|_\infty = 1} \|\Sigma_{SS}^{-1} v\|_\infty \|\Sigma_{SS^c} \widehat{\beta}^{(P1)}\|_\infty \overset{(10)}{\le} K(S) \cdot \begin{cases} \mu(S) \|\widehat{\beta}^{(P1)}\|_1 & \text{for (i)}, \\ \mu_+(S) \|\widehat{\beta}^{(P1)}\|_\infty & \text{for (ii)}. \end{cases} \tag{14}$$

**Step 3:** *Putting together the pieces.* The two random terms $M$ and $\overline{M}$ are maxima of a finite collection of sub-Gaussian random variables, which can be controlled using standard techniques. Since

$\|Z_j\|_2 \leq \|X_j\|_2$ and $\|e_j^\top \Sigma_{SS}^{-1} X_S^\top / \sqrt{n}\|_2 \leq \{\phi_{\min}(S)\}^{-1/2}$ for all $j$, the sub-Gaussian parameters of these collections are upper bounded by $\sigma/\sqrt{n}$ and $\sigma/(\{\phi_{\min}(S)\}^{1/2}\sqrt{n})$, respectively. It follows that the two events $\{M \leq 2\sigma\sqrt{\frac{2\log p}{n}}\}$ and $\{\overline{M} \leq \frac{2\sigma}{\{\phi_{\min}(S)\}^{1/2}}\sqrt{\frac{2\log p}{n}}\}$ both hold with probability no less than $1 - 10/p$, cf. supplement. Subsequently, we work conditional on these two events. For the choice of $\lambda$ made for (i) and (ii), respectively, it follows that

$$\|\beta^* - \bar{\beta}^{(P2)}\|_\infty \leq \frac{2\sigma}{\{\phi_{\min}(S)\}^{1/2}}\sqrt{\frac{2\log p}{n}} + \lambda K(S) \cdot \begin{cases} \mu(S) & \text{for (i),} \\ \mu_+(S) & \text{for (ii),} \end{cases}$$

and hence, using the lower bound on $\beta_{\min}(S)$, that $\bar{\beta}^{(P2)} = \widehat{\beta}_S \succ 0$ and thus also that $\widehat{\beta}^{(P1)} = \widehat{\beta}_{S^c}$. Subsequent thresholding with the respective choices made for $\lambda$ yields the assertion. □

In the sequel, we apply Theorem 2 to specific classes of designs commonly studied in the literature, for which thresholded NNLS achieves an $\ell_\infty$-error of the optimal order $O(\sqrt{\log(p)/n})$. We here only provide sketches, detailed derivations are relegated to the supplement.

**Example 1: Power decay.** Let the entries of the Gram matrix $\Sigma$ be given by $\sigma_{jk} = \rho^{|j-k|}$, $1 \leq j, k \leq p$, $0 \leq \rho < 1$, so that the $\{X_j\}_{j=1}^p$ form a Markov random field in which $X_j$ is conditionally independent of $\{X_k\}_{k \notin \{j-1,j,j+1\}}$ given $\{X_{j-1}, X_{j+1}\}$, cf. [24]. The conditional independence structure implies that all entries of $Z^\top Z$ are non-negative, such that, using the definition of $\widehat{\omega}(S)$,

$$\widehat{\omega}(S) \geq \min_{1 \leq j \leq p-s} \min_{v \succeq 0, \|v\|_\infty = 1} \left|\frac{1}{n}Z_j^\top Z v\right| = \min_{1 \leq j \leq (p-s)} \frac{1}{n}(Z^\top Z)_{jj} + \frac{1}{n}\sum_{k \neq j} \min\{(Z^\top Z)_{jk}, 0\},$$

the sum on the r.h.s. vanishes, thus one computes $\widehat{\omega}(S) \geq \min_{1 \leq j \leq (p-s)} \frac{1}{n}(Z^\top Z)_{jj} \geq 1 - \frac{2\rho^2}{1+\rho^2}$ for all $S$. For the remaining constants in (10), one can show that $\Sigma_{SS}^{-1}$ is a band matrix of bandwidth no more than 3 for all choices of $S$ such that $\phi_{\min}(S)$ and $K(S)$ are uniformly lower and upper bounded, respectively, by constants depending on $\rho$ only. By the geometric series formula, $\mu_+(S) \leq \frac{\rho}{1-\rho}$. In total, for a constant $C_\rho > 0$ depending on $\rho$ only, one obtains an $\ell_\infty$-error of the form

$$\|\widehat{\beta}(\lambda) - \beta^*\|_\infty \leq C_\rho \sigma \sqrt{2\log(p)/n}. \tag{15}$$

**Example 2: Equi-correlation.** Suppose that $\sigma_{jk} = \rho$, $0 < \rho < 1$, for all $j \neq k$, and $\sigma_{jj} = 1$ for all $j$. For any $S$, one computes that the matrix $\frac{1}{n}Z^\top Z$ is of the same regular structure with diagonal entries all equal to $1 - \delta$ and off-diagonal entries all equal to $\rho - \delta$, where $\delta = \rho^2 s/(1 + (s-1)\rho)$. Therefore, using (8), the separating hyperplane constant (7) can be computed in closed form:

$$\widehat{\tau}^2(S) = \frac{(1-\rho)\rho}{(s-1)\rho + 1} + \frac{1-\rho}{p-s} = O(s^{-1}). \tag{16}$$

Arguing as in (12) in the proof of Theorem 2, this allows one to show that with high probability,

$$\|\widehat{\beta}_{S^c}\|_1 \leq \frac{2\sigma\sqrt{2\log(p)/n}}{\widehat{\tau}^2(S)} \leq \frac{((s-1)\rho + 1)2\sigma\sqrt{2\log(p)/n}}{(1-\rho)\rho}. \tag{17}$$

On the other hand, using the same reasoning as in Example 1, $\widehat{\omega}(S) \geq 1 - \delta = c_\rho > 0$, say. Choosing the threshold $\lambda = \frac{2\sigma}{\widehat{\omega}(S)}\sqrt{\frac{2\log p}{n}}$ as in part (ii) of Theorem 2 and combining the strong $\ell_1$-bound (17) on the off-support coefficients with a slight modification of the bound (14) together with $\phi_{\min}(S) = 1 - \rho$ yields again the desired optimal bound of the form (15).

**Random designs.** So far, the design matrix $X$ has been assumed to be fixed. Consider the following ensemble of random matrices

$\text{Ens}_+ = \{X = (x_{ij}), \{x_{ij},\ 1 \leq i \leq n,\ 1 \leq j \leq p\}$ i.i.d. from a sub-Gaussian distribution on $\mathbb{R}_+\}$.

Among others, the class of sub-Gaussian distributions on $\mathbb{R}_+$ encompasses all distributions on a bounded set on $\mathbb{R}_+$, e.g. the family of beta distributions (with the uniform distribution as special case) on $[0, 1]$, Bernoulli distributions on $\{0, 1\}$ or more generally distributions on counts

$\{0, 1, \ldots, K\}$, for some positive integer $K$. The ensemble $\text{Ens}_+$ is well amenable to analysis, since after suitable re-scaling the corresponding population Gram matrix $\Sigma^* = \mathbf{E}[\frac{1}{n} X^\top X]$ has equi-correlation structure (Example 2): denoting the mean of the entries and their squares by $\mu$ and $\mu_2$, respectively, we have $\Sigma^* = (\mu_2 - \mu^2)I + \mu^2 \mathbf{1}\mathbf{1}^\top$ such that re-scaling by $1/\sqrt{\mu_2}$ leads to equi-correlation with $\rho = \mu^2/\mu_2$. As shown above, the incoherence constant $\widehat{\tau}^2(S)$, which gives rise to a strong bound on $\|\widehat{\beta}_{S^c}\|_1$, scales favourably and can be computed in closed form. For random designs from $\text{Ens}_+$, one additionally has to take into account the deviation between $\Sigma$ and $\Sigma^*$. Using tools from random matrix theory, we show that the deviation is moderate, of the order $O(\sqrt{\log(p)/n})$.

**Theorem 3.** *Let $X$ be a random matrix from $\text{Ens}_+$, scaled s.t. $\mathbf{E}\left[\frac{1}{n} X^\top X\right] = \rho I + (1-\rho)\mathbf{1}\mathbf{1}^\top$ for some $\rho \in (0, 1)$. Fix an $S \subset \{1, \ldots, p\}$, $|S| \leq s$. Then there exists constants $c, c_1, c_2, c_3, C, C' > 0$ such that for all $n \geq C \log(p)s^2$,*

$$\widehat{\tau}^2(S) \geq cs^{-1} - C'\sqrt{\log(p)/n}$$

*with probability no less than $1 - 3/p - \exp(-c_1 n) - 2\exp(-c_2 \log p) - \exp(-c_3 \log^{1/2}(p)s)$.*

## 4 Experiments

**Setup.** We randomly generate data $y = X\beta^* + \varepsilon$, where $\varepsilon$ has i.i.d. standard Gaussian entries. We consider two choices for the design $X$. For one set of experiments, the rows of $X$ are drawn i.i.d. from a Gaussian distribution whose covariance matrix has the power decay structure of Example 1 with parameter $\rho = 0.7$. For the second set, we pick a representative of the class $\text{Ens}_+$ by drawing each entry of $X$ uniformly from $[0, 1]$ and re-scaling s.t. the population Gram matrix $\Sigma^*$ has equi-correlation structure with $\rho = 3/4$. The target $\beta^*$ is generated by selecting its support $S$ uniformly at random and then setting $\beta_j^* = b \cdot \beta_{\min}(S)(1 + U_j)$, $j \in S$, where $\beta_{\min}(S) = C_\rho \sigma \sqrt{2 \log(p)/n}$, using upper bounds for the constant $C_\rho$ as used for Examples 1 and 2; the $\{U_j\}_{j \in S}$ are drawn i.i.d. uniformly from $[0, 1]$, and $b$ is a parameter controlling the signal strength. The experiments can be divided into two parts. In the first part, the parameter $b$ is kept fixed while the aspect ratio $p/n$ of $X$ and the fraction of sparsity $s/n$ vary. In the second part, $s/n$ is fixed to $0.2$, while $p/n$ and $b$ vary. When not fixed, $s/n \in \{0.05, 0.1, 0.15, 0.2, 0.25, 0.3\}$. The grid used for $b$ is chosen specific to the designs, calibrated such that the sparse recovery problems are sufficiently challenging. For the design from $\text{Ens}_+$, $p/n \in \{2, 3, 5, 10\}$, whereas for power decay $p/n \in \{1.5, 2, 2.5, 3, 3.5, 4\}$, for reasons that become clear from the results. Each configuration is replicated 100 times for $n = 500$.

**Comparison.** Across these runs, we compare the probability of 'success' of thresholded NNLS (tNNLS), non-negative lasso ($\text{NN}\ell_1$), thresholded non-negative lasso ($\text{tNN}\ell_1$) and orthogonal matching pursuit (OMP, [22, 23]). For a regularization parameter $\mu \geq 0$, $\text{NN}\ell_1$ is defined as a minimizer $\widehat{\beta}(\mu)$ of $\min_{\beta \succeq 0} \frac{1}{n} \|y - X\beta\|_2^2 + \mu \mathbf{1}^\top \beta$. We also compare against the ordinary lasso (replacing $\mathbf{1}^\top \beta$ by $\|\beta\|_1$ and removing the non-negativity constraint); since its performance is mostly nearly equal, partially considerably worse than that of its non-negative counterpart (see the bottom right panel of Figure 4 for an example), the results are not shown in the remaining plots for the sake of better readability. 'Success' is defined as follows. For tNNLS, we have 'success' if $\min_{j \in S} \widehat{\beta}_j > \max_{j \in S^c} \widehat{\beta}_j$, i.e. there exists a threshold that permits support recovery. For $\text{NN}\ell_1$, we set $\widehat{\mu} = 2\|X^\top \varepsilon/n\|_\infty$, which is the empirical counterpart to $\mu_0 = 2\sqrt{2 \log(p)/n}$, the choice for the regularization parameter advocated in [14] to achieve the optimal rate for estimating $\beta^*$ in the $\ell_2$-norm, and compute the whole set of solutions $\{\widehat{\beta}(\mu), \ \mu \geq \widehat{\mu}\}$ using the non-negative lasso modification of LARS [26] and check whether the sparsity pattern of *one* of these solutions recovers $S$. For $\text{tNN}\ell_1$, we inspect $\{\widehat{\beta}(\mu) : \mu \in [\mu_0 \wedge \widehat{\mu}, \mu_0 \vee \widehat{\mu}]\}$ and check whether $\min_{j \in S} \widehat{\beta}_j(\mu) > \max_{j \in S^c} \widehat{\beta}_j(\mu)$ holds for *one* of these solutions. For OMP, we check whether the support $S$ is recovered in the first $s$ steps. Note that, when comparing tNNLS and $\text{tNN}\ell_1$, the lasso is given an advantage, since we optimize over a range of solutions.

*Remark:* We have circumvented the choice of the threshold $\lambda$, which is crucial in practice. In a specific application [5] the threshold is chosen in a signal-dependent way allowing domain experts to interpret $\lambda$ as signal-to-noise ratio. Alternatively, one can exploit that under the conditions of Theorem 2, the $s$ largest coefficients of $\widehat{\beta}$ are those of the support. Given a suitable data-driven estimate for $s$ e.g. that proposed in [25], $\lambda$ can be chosen automatically.

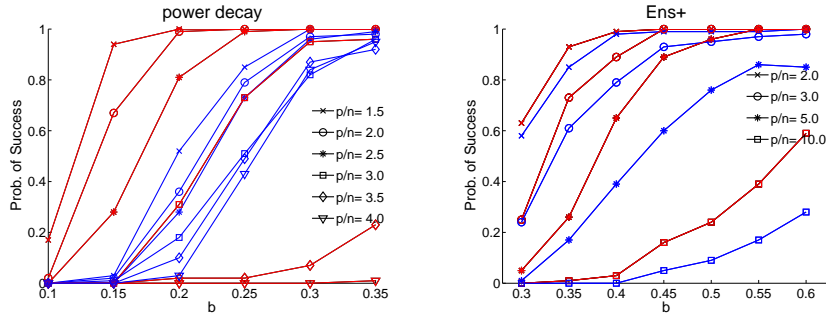

Figure 3: Comparison of thresholded NNLS (red) and thresholded non-negative lasso (blue) for the experiments with constant $s/n$, while $b$ (abscissa) and $p/n$ (symbols) vary.

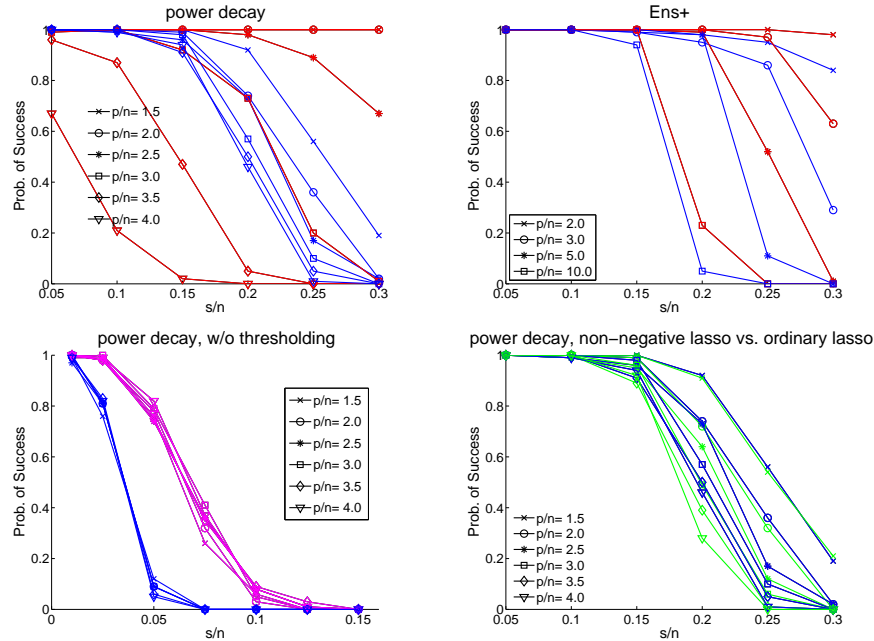

Figure 4: **Top:** Comparison of thresholded NNLS (red) and the thresholded non-negative lasso (blue) for the experiments with constant $b$, while $s/n$ (abscissa) and $p/n$ (symbols) vary. **Bottom left**: Non-negative lasso *without* thresholding (blue) and orthogonal matching pursuit (magenta). **Bottom right**: Thresholded non-negative lasso (blue) and thresholded ordinary lasso (green).

**Results.** The approaches NN$\ell_1$ and OMP are not competitive − both work only with rather moderate levels of sparsity, with a breakdown at $s/n = 0.15$ for power decay as displayed in the bottom left panel of Figure 4. For the second design, the results are even worse. This is in accordance with the literature where thresholding is proposed as remedy [17, 18, 19]. Yet, for a wide range of configurations, tNNLS visibly outperforms tNN$\ell_1$, a notable exception being power decay with larger values for $p/n$. This is in contrast to the design from Ens$_+$, where even $p/n = 10$ can be handled. This difference requires further research.

**Conclusion.** To deal with higher levels of sparsity, thresholding seems to be inevitable. Thresholding the biased solution obtained by $\ell_1$-regularization requires a proper choice of the regularization parameter and is likely to be inferior to thresholded NNLS with regard to the detection of small signals. The experimental results provide strong support for the central message of the paper: even in high-dimensional, noisy settings, non-negativity constraints can be unexpectedly powerful when interacting with 'self-regularizing 'properties of the design. While this has previously been observed empirically, our results provide a solid theoretical understanding of this phenomenon. A natural question is whether this finding can be transferred to other kinds of 'simple constraints' (e.g. box constraints) that are commonly imposed.

# References

[1] Y. Lin, D. Lee, and L. Saul. Nonnegative deconvolution for time of arrival estimation. In *ICASSP*, 2004.

[2] J. Bardsley and J. Nagy. Covariance-preconditioned iterative methods for nonnegatively constrained astronomical imaging. *SIAM Journal on Matrix Analysis and Applications*, 27:1184–1198, 2006.

[3] A. Szlam and. Z. Guo and S. Osher. A split Bregman method for non-negative sparsity penalized least squares with applications to hyperspectral demixing. In *IEEE International Conference on Image Processing*, 2010.

[4] L. Li and T. Speed. Parametric deconvolution of positive spike trains. *The Annals of Statistics*, 28:1279–1301, 2000.

[5] M. Slawski and M. Hein. Sparse recovery for Protein Mass Spectrometry data. In *NIPS workshop on practical applications of sparse modelling*, 2010.

[6] D. Donoho, I. Johnstone, J. Hoch, and A. Stern. Maximum entropy and the nearly black object. *Journal of the Royal Statistical Society Series B*, 54:41–81, 1992.

[7] D. Chen and R. Plemmons. Nonnegativity constraints in numerical analysis. In *Symposium on the Birth of Numerical Analysis*, 2007.

[8] A. Bruckstein, M. Elad, and M. Zibulevsky. On the uniqueness of nonnegative sparse solutions to underdetermined systems of equations. *IEEE Transactions on Information Theory*, 54:4813–4820, 2008.

[9] D. Donoho and J. Tanner. Counting the faces of randomly-projected hypercubes and orthants, with applications. *Discrete and Computational Geometry*, 43:522–541, 2010.

[10] M. Wang and A. Tang. Conditions for a Unique Non-negative Solution to an Underdetermined System. In *Proceedings of Allerton Conference on Communication, Control, and Computing*, 2009.

[11] M. Wang, W. Xu, and A. Tang. A unique nonnegative solution to an undetermined system: from vectors to matrices. *IEEE Transactions on Signal Processing*, 59:1007–1016, 2011.

[12] C. Liew. Inequality Constrained Least-Squares Estimation. *Journal of the American Statistical Association*, 71:746–751, 1976.

[13] R. Tibshirani. Regression shrinkage and variable selection via the lasso. *Journal of the Royal Statistical Society Series B*, 58:671–686, 1996.

[14] S. van de Geer and P. Bühlmann. On the conditions used to prove oracle results for the Lasso. *The Electronic Journal of Statistics*, 3:1360–1392, 2009.

[15] P. Zhao and B. Yu. On model selection consistency of the lasso. *Journal of Machine Learning Research*, 7:2541–2567, 2006.

[16] M. Wainwright. Sharp thresholds for noisy and high-dimensional recovery of sparsity using $\ell_1$-constrained quadratic programming (Lasso). *IEEE Transactions on Information Theory*, 55:2183–2202, 2009.

[17] N. Meinshausen and B. Yu. Lasso-type recovery of sparse representations for high-dimensional data. *The Annals of Statistics*, 37:246–270, 2009.

[18] T. Zhang. Some Sharp Performance Bounds for Least Squares Regression with $L_1$ Regularization. *The Annals of Statistics*, 37:2109–2144, 2009.

[19] S. Zhou. Thresholding procedures for high dimensional variable selection and statistical estimation. In *NIPS*, 2009.

[20] E. Greenshtein and Y. Ritov. Persistence in high-dimensional linear predictor selection and the virtue of overparametrization. *Bernoulli*, 6:971–988, 2004.

[21] D. Donoho and J. Tanner. Sparse nonnegative solution of underdetermined linear equations by linear programming. *Proceedings of the National Academy of Science*, 102:9446–9451, 2005.

[22] J. Tropp. Greed is good: Algorithmic results for sparse approximation. *IEEE Transactions on Information Theory*, 50:2231–2242, 2004.

[23] T. Zhang. On the Consistency of Feature Selection using Greedy Least Squares Regression. *Journal of Machine Learning Research*, 10:555–568, 2009.

[24] H. Rue and L. Held. *Gaussian Markov Random Fields*. Chapman and Hall/CRC, Boca Raton, 2001.

[25] C. Genovese, J. Jin, and L. Wasserman. Revisiting Marginal Regression. Technical report, Carnegie Mellon University, 2009. http://arxiv.org/abs/0911.4080.

[26] B. Efron, T. Hastie, I. Johnstone, and R. Tibshirani. Least Angle Regression. *The Annals of Statistics*, 32:407–499, 2004.

